# Efficient estimation of hidden state dynamics from spike trains

**Márton G. Danóczy**
Inst. for Theoretical Biology
Humboldt University, Berlin
Invalidenstr. 43
10115 Berlin, Germany
m.danoczy@biologie.hu-berlin.de

**Richard H. R. Hahnloser**
Inst. for Neuroinformatics
UNIZH / ETHZ
Winterthurerstrasse 190
8057 Zurich, Switzerland
rich@ini.phys.ethz.ch

## Abstract

Neurons can have rapidly changing spike train statistics dictated by the underlying network excitability or behavioural state of an animal. To estimate the time course of such state dynamics from single- or multiple neuron recordings, we have developed an algorithm that maximizes the likelihood of observed spike trains by optimizing the state lifetimes and the state-conditional *interspike-interval* (ISI) distributions. Our non-parametric algorithm is free of time-binning and spike-counting problems and has the computational complexity of a Mixed-state Markov Model operating on a state sequence of length equal to the total number of recorded spikes. As an example, we fit a two-state model to paired recordings of premotor neurons in the sleeping songbird. We find that the two state-conditional ISI functions are highly similar to the ones measured during waking and singing, respectively.

## 1 Introduction

It is well known that neurons can suddenly change firing statistics to reflect a macroscopic change of a nervous system. Often, firing changes are not accompanied by an immediate behavioural change, as is the case, for example, in paralysed patients, during sleep [1], during covert discriminative processing [2], and for all in-vitro studies [3]. In all of these cases, changes in some hidden macroscopic state can only be detected by close inspection of single or multiple spike trains. Our goal is to develop a powerful, but computationally simple tool for point processes such as spike trains. From spike train data, we want to the extract continuously evolving hidden variables, assuming a discrete set of possible states.

Our model for classifying spikes into discrete hidden states is based on three assumptions:

1. Hidden states form a continuous-time Markov process and thus have exponentially distributed lifetimes

2. State switching can occur only at the time of a spike (where there is observable evidence for a new state).

3. In each of the hidden states, spike trains are generated by mutually independent renewal processes.

**1.** For a continuous-time Markov process, the probability of staying in state $S = i$ for a time interval $T > t$ is given by $P_i(t) = \exp(-r_i t)$, where $r_i$ is the escape rate (or hazard rate) of state $i$. The mean lifetime $\tau_i$ is defined as the inverse of the escape rate, $\tau_i = 1/r_i$.

As a corollary, it follows that the probability of staying in state $i$ for a particular duration equals the probability of surviving for a fraction of that duration times the probability of surviving for the remaining time, i.e., the state survival probability $P_i(t)$ satisfies the product identity

$$P_i(t_1 + t_2) = P_i(t_1)P_i(t_2).$$

**2.** According to the second assumption, state switching can occur at any spike, irrespective of which neuron fired the spike. In the following, we shall refer to a spike fired by any of the neurons as an *event* (where state switching might occur). Note that if two (or more) neurons happen to fire a spike at exactly the same time, the respective spikes are regarded as two (or more) distinct events. The collection of event times is denoted by $t_e$.

Combining the first two assumptions, we formulate the hidden state sequence at the events (i.e. observation points) as a non-homogeneous discrete Markov chain. Accordingly, the probability of remaining in state $i$ for the duration of the *interevent-interval* (IEI) $\Delta t_e = t_e - t_{e-1}$ is given by the state survival probability $P_i(\Delta t_e)$. The probability to change state is then $1 - P_i(\Delta t_e)$.

**3.** In each state $i$, the spike trains are assumed to be generated by a renewal process that randomly draws *interspike-intervals* (ISIs) $t$ from a *probability density function* (pdf) $h_i(t)$. Because every IEI is only a fraction of an ISI, instead of working with ISI distributions, we use an equivalent formulation based on the *conditional intensity function* (CIF) $\lambda_i(\varphi)$ [4]. The CIF, also called hazard function in reliability theory, is a generalization of the Poisson firing rate. It is defined as the probability density of spiking in the time interval $[\varphi, \varphi + \mathrm{d}t]$, given that no spike has occurred in the interval $[0, \varphi)$ since the last spike. In the following, the variable $\varphi$, i.e. the time that has elapsed since the last spike, shall be referred to as *phase* [5]. Using the CIF, the ISI pdf can be expressed by the fundamental equation of renewal theory,

$$h_i(t) = \exp\left(-\int_0^t \lambda_i(\varphi)\,\mathrm{d}\varphi\right)\lambda_i(t). \tag{1}$$

At each event $e$, we observe the phase trajectory of every neuron traced out since the last event. It is clear that in multiple electrode recordings the phase trajectories between events are not independent, since they have to start where the previous trajectory ended. Therefore, our model violates the observation independence assumption of standard *Hidden Markov Models* (HMMs). Our model is, in formal terms, a mixed-state Markov model [6], with the architecture of a double-chain [7]. Such models are generalizations of HMMs in that the observable outputs may not only be dependent on the current hidden state, but also on past observations (formally, the mixed state is formed by combining the hidden and observable states).

In our model, hidden state transition probabilities are characterized by the escape rates $r_i$ and observable state transition probabilities by the CIFs $\lambda_i^n$ for neuron $n$ in hidden state $i$. Our goal is to find a set $\Psi$ of model parameters, such that the likelihood

$$\Pr\{\mathbf{O}|\Psi\} = \sum_{\mathbf{S}\in\mathbb{G}} \Pr\{\mathbf{S},\mathbf{O}|\Psi\}$$

of the observation sequence $\mathbf{O}$ is maximized.

As a first step, we will derive an expression for the combined likelihood $\Pr\{\mathbf{S},\mathbf{O}|\Psi\}$. Then, we will apply the *expectation maximization* (EM) algorithm to find the optimal parameter set.

## 2 Transition probabilities

The mixed state at event $e$ shall be composed of the hidden state $S_e$ and the observable outputs $O_e^n$ (for neurons $n \in \{1,\dots,N\}$).

**Hidden state transitions**  In classical mixed-state Markov models, the hidden state transition probabilities are constant. In our model, however, we describe time as a continuous quantity and observe the system whenever a spike occurs, thus in non-equidistant intervals. Consequently, hidden state transitions depend explicitly on the elapsed time since the last observation, i.e., on the IEIs $\Delta t_e$. The transition probability $a_{ij}(\Delta t_e)$ from hidden state $i$ to hidden state $j$ is then given by

$$a_{ij}(\Delta t_e) = \begin{cases} \exp(-r_j \Delta t_e) & \text{if } i = j, \\ [1 - \exp(-r_j \Delta t_e)]\, g_{ij} & \text{otherwise,} \end{cases} \tag{2}$$

where $g_{ij}$ is the conditional probability of making a transition from state $i$ into a new state $j$, given that $j \neq i$. Thus, $g_{ij}$ has to satisfy the constraint $\sum_j g_{ij} = 1$, with $g_{ii} = 0$.

**Observable state transitions**  The observation at event $e$ is defined as $O_e = \{\Phi_e^n, v_e\}$, where $v_e$ contains the index of the neuron that has triggered event $e$ by emitting a spike, and $\Phi_e^n = (\inf \Phi_e^n, \sup \Phi_e^n]$ is the phase interval traced out by neuron $n$ since its last spike. Observations form a cascade. After a spike, the phase of the respective neuron is immediately reset to zero. The interval's bounds are thus defined by

$$\sup \Phi_e^n = \inf \Phi_e^n + \Delta t_e \quad \text{and} \quad \inf \Phi_e^n = \begin{cases} 0 & \text{if } v_{e-1} = n, \\ \sup \Phi_{e-1}^n & \text{otherwise.} \end{cases}$$

The observable transition probability $p_i(O_e) = \Pr\{O_e \mid O_{e-1}, S_e = i\}$ is the probability of observing output $O_e$, given the previous output $O_{e-1}$ and the current hidden state $S_e$. With our independence assumption (3.), we can give its density as the product of every neuron's probability of having survived the respective phase interval $\Phi_e^n$ that it has traced out since its last spike, multiplied by the spiking neuron's firing rate (compare equation 1):

$$p_i(O_e) = \left[ \prod_n \exp\left( -\int_{\Phi_e^n} \lambda_i^n(\varphi)\, d\varphi \right) \right] \lambda_i^{v_e}(\sup \Phi_e^{v_e}). \tag{3}$$

Note that in case of a single neuron recording, this reduces to the ISI pdf.

To give a closed form of the observable transition pdf, several approaches are thinkable. Here, for the sake of flexibility and computational simplicity, we approximate the CIF $\lambda_i^n$ for neuron $n$ in state $i$ by a step function, assuming that its value is constant inside small, arbitrarily spaced bins $B^n(b), b \in \{1,\dots,N_{\text{bins}}^n\}$. That is, $\lambda_i^n(\varphi) \approx \ell_i^n(b), \forall \varphi \in B^n(b)$.

In order to use the discretized CIFs $\ell_i^n(b)$, we also discretize $\Phi_e^n$: the fractions $f_e^n(b) \in [0,1]$ represent how much of neuron $n$'s phase bin $B^n(b)$ has been traced out since the last event. For example, if event $e-1$ happened in the middle of neuron $n$'s phase bin 2 and event $e$ happened ten percent into its phase bin 4, then $f_e^n(2) = 0.5$, $f_e^n(3) = 1$, and $f_e^n(4) = 0.1$, whereas $f_e^n(i) = 0$ for other $i$, Figure 1.

Making use of these discretizations, the integral in equation 3 is approximated by a sum:

$$p_i(O_e) \approx \left[ \prod_n \exp\left( -\sum_{b=1}^{N_{\text{bins}}^n} f_e^n(b)\, \ell_i^n(b)\, \|B^n(b)\| \right) \right] \lambda_i^{v_e}(\sup \Phi_e^{v_e}), \tag{4}$$

with $\|B^n(b)\|$ denoting the width of neuron $n$'s phase bin $b$.

Equations 2 and 4 fully describe transitions in our mixed-state Markov model. Next, we apply the EM algorithm to find optimal values of the escape rates $r_i$, the conditional hidden state transition probabilities $g_{ij}$ and the discretized CIFs $\ell_i^n(b)$, given a set of spike trains.

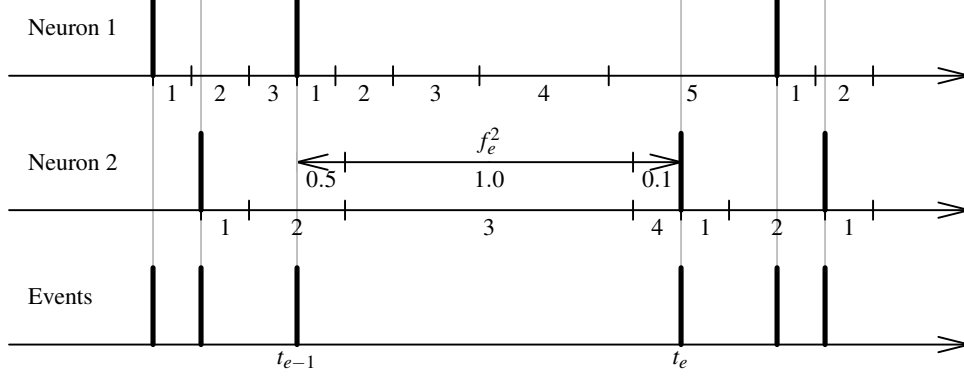

Figure 1: Two spike trains are combined to form the event train shown in the bottom row. The phase bins are shown below the spike trains, they are labelled with the corresponding bin number. As an example, for the second neuron, the fractions $f_e^2(b)$ of its phase bins that have been traced out since event $e-1$ are indicated by the horizontal arrow. They are nonzero for $b = 2, 3$, and 4.

## 3   Parameter estimation

Our goal is to find model parameters $\Psi = \{r_i, g_{ij}, \ell_i^n(b)\}$, such that the likelihood $\Pr\{\mathbf{O}|\Psi\}$ of observation sequence $\mathbf{O}$ is maximized. According to the EM algorithm, we can find such values by iterating over models

$$\Psi^{\text{new}} = \arg\max_{\psi} \sum_{\mathbf{S} \in \mathfrak{S}} \Pr\{\mathbf{S}|\mathbf{O}, \Psi^{\text{old}}\} \ln(\Pr\{\mathbf{S}, \mathbf{O}|\psi\}), \tag{5}$$

where $\mathfrak{S}$ is the set of all possible hidden state sequences. The product of equations 2 and 4 over all events is proportional to the combined likelihood $\Pr\{\mathbf{S}, \mathbf{O}|\psi\}$:

$$\Pr\{\mathbf{S}, \mathbf{O}|\psi\} \sim \prod_e a_{S_{e-1} S_e}(\Delta t_e) \, p_{S_e}(O_e).$$

Because of the logarithm in equation 5, the maximization over escape rates can be separated from the maximization over conditional intensity functions. We define the abbreviations $\xi_{ij}(e) = \Pr\{S_{e-1} = i, S_e = j | \mathbf{O}, \Psi^{\text{old}}\}$ and $\gamma_i(e) = \Pr\{S_e = i | \mathbf{O}, \Psi^{\text{old}}\}$ for the posterior probabilities appearing in equation 5. In practice, both expressions are computed in the expectation step by the classic forward-backward algorithm [8], using equations 2 and 4 as the transition probabilities. With the abbreviations defined above, equation 5 is split to

$$r_j^{\text{new}} = \arg\max_r \left( \sum_e \xi_{jj}(e)(-r\Delta t_e) + \sum_{e, i \neq j} \xi_{ij}(e) \ln[1 - \exp(-r\Delta t_e)] \right) \tag{6}$$

$$\ell_i^n(b)^{\text{new}} = \arg\max_\ell \left( -\ell \sum_e \gamma_i(e) f_e^n(b) \|B^n(b)\| + \ln\ell \sum_{\substack{e: v_e = n \, \wedge \\ \sup \Phi_e^n \in B^n(b)}} \gamma_i(e) \right) \tag{7}$$

$$g_{ij}^{\text{new}} = \arg\max_g \left( \ln g \sum_e \xi_{ij}(e) \right) \text{ with } g_{ii}^{\text{new}} = 0 \text{ and } \sum_j g_{ij}^{\text{new}} = 1. \tag{8}$$

In order to perform the maximization in equation 6, we compute its derivative with respect to $r$ and set it to zero:

$$0 = \sum_e \xi_{jj}(e)\Delta t_e + \left( \Delta t_e - \frac{\Delta t_e}{1 - \exp\left(-r_j^{\text{new}}\Delta t_e\right)} \right) \sum_{i \neq j} \xi_{ij}(e)$$

This equation cannot be solved analytically, but being just a one dimensional optimization problem, a solution can be found using numerical methods, such as the Levenberg-Marquardt algorithm. The singularity in case of $\Delta t_e = 0$, which arises when two or more spikes occur at the same time, needs the special treatment of replacing the respective fraction by its limit: $1/r_i^{\mathrm{new}}$.

To obtain the reestimation formula for the discretized CIFs, equation 7's derivative with respect to $\ell$ is set to zero. The result can be solved directly and yields

$$\ell_i^n(b)^{\mathrm{new}} = \sum_{\substack{e: v_e = n \,\wedge \\ \sup \Phi_e^n \in B^n(b)}} \gamma_i(e) \left/ \sum_e \gamma_i(e)\, f_e^n(b)\, \|B^n(b)\|\right..$$

Finally, to obtain the reestimation formula for the conditional hidden state transition probabilities $g_{ij}$, we solve equation 8 using Lagrange multipliers, resulting in

$$g_{i \neq j}^{\mathrm{new}} = \sum_e \xi_{ij}(e) \left/ \sum_{e, k \neq i} \xi_{ik}(e)\right..$$

# 4   Application to spike trains from the sleeping songbird

We have applied our model to spike train data from sleeping songbirds [9]. It has been found that during sleep, neurons in vocal premotor area RA exhibit spontaneous activity that at times resembles premotor activity during singing [10, 9].

We train our model on the spike train of a single RA neuron in the sleeping bird with $N_{\mathrm{bins}} = 100$, where the first bin extends from the sample time to 1ms and the consecutive 99 steps are logarithmically spaced up to the largest ISI. After convergence, we find that the ISI pdfs associated with the two hidden states qualitatively agree with the pdfs recorded in the awake non-singing bird and the awake singing bird, respectively, Figure 2. ISI pdfs were derived from the CIFs by using equation 1. For the state-conditional ISI histograms we first ran the Viterbi algorithm to find the most likely hidden-state sequence and then sorted spikes into two groups, for which the ISIs histograms were computed.

We find that sleep-related activity in the RA neuron of Figure 2 is best described by random switching between a singing-like state of lifetime $\tau_1 = 1.18s \pm 0.38s$ and an awake, non-singing-like state of lifetime $\tau_2 = 2.26s \pm 0.42s$. Standard deviations of lifetime estimates were computed by dividing the spike train into 30 data windows of 10$s$ duration each and computing the Jackknife variance [11] on the truncated spike trains. The difference between the singing-like state in our model and the true singing ISI pdf shown in Figure 2 is more likely due to generally reduced burst rates during sleep, rather than to a particularity of the examined neuron.

Next we applied our model to simultaneous recordings from pairs of RA neurons. By fitting two separate models (with identical phase binning) to the two spike trains, and after running the Viterbi algorithm to find the most likely hidden state sequences, we find good agreement between the two sequences, Figure 3 (top row) and 4c. The correspondence of hidden state sequences suggests a common network mechanism for the generation of the singing-like states in both neurons. We thus applied a single model to both spike trains and found again good agreement with hidden-state sequences determined for the separate models, Figure 3 (bottom row) and 4f. The lifetime histograms for both states look approximatively exponential, justifying our assumption for the state dynamics, Figure 4g and h.

For the model trained on neuron one we find lifetimes $\tau_1 = 0.63s \pm 0.37s$ and $\tau_2 = 1.71s \pm 0.45s$, and for the model trained on neuron two we find $\tau_1 = 0.42s \pm 0.11s$ and $\tau_2 = 1.23s \pm 0.17s$. For the combined model, lifetimes are $\tau_1 = 0.58s \pm 0.25s$ and

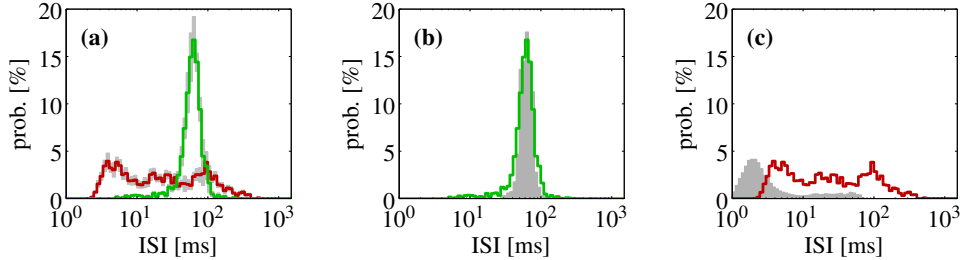

Figure 2: **(a)**: The two state-conditional ISI histograms of an RA neuron during sleep are shown by the red and green curves, respectively. Gray patches represent Jackknife standard deviations. **(b)**: After waking up the bird by pinching his tail, the new ISI histogram shown by the gray area becomes almost indistinguishable from the ISI histogram of state 1 (green line). **(c)**: In comparison to the average ISI histogram of many RA neurons during singing (shown by the gray area, reproduced from [12]), the ISI histogram corresponding to state 2 (red line) is shifted to the right, but looks otherwise qualitatively similar.

$\tau_2 = 1.13s \pm 0.15s$. Thus, hidden-state switching seems to occur more frequently in the combined model. The reason for this increase might be that evidence for the song-like state appears more frequently with two neurons, as a single neuron might not be able to indicate song-like firing statistics with high temporal fidelity.

We have also analysed the correlations between state dynamics in the different models. The hidden state function $S(t)$ is a binary function that equals one when in hidden state 1 and zero when in state 2. For the case where we modelled the two spike trains separately, we have two such hidden state functions, $S^1(t)$ for neuron one and $S^2(t)$ for neuron two. We find that all correlation functions $C_{SS^1}(t)$, $C_{SS^2}(t)$, and $C_{S^1S^2}(t)$, have a peak at zero time lag, with a high peak correlation of about 0.7, Figure 4c and f (the correlation function is defined as the cross-covariance function divided by the autocovariance functions).

We tested whether our model is a good generative model for the observed spike trains by applying the time rescaling theorem, after which the ISIs of a good generative model with known CIFs should reduce to a Poisson process with unit rate, which, after another transformation, should lead to a uniform probability density in the interval $(0,1)$ [4]. Performing this test, we found that the transformed ISI densities of the combined model are uniform, thus validating our model (95% Kolmogorov-Smirnov test, Figure 4i).

## 5 Discussion

We have presented a mixed-state Markov model for point processes, assuming generation by random switching between renewal processes. Our algorithm is suited for systems in which neurons make discrete state transitions simultaneously. Previous attempts of fitting spike train data with Markov models exhibited weaknesses due to time binning. With large time bins and the number of spikes per bin treated as observables [13, 14], state transitions can only be detected when they are accompanied by firing rate changes. In our case, RA neurons have a roughly constant firing rate throughout the entire recording, and so such approaches fail.

We were able to model the hidden states in continuous time, but had to bin the ISIs in order to deal with limited data. In principle, the algorithm can operate on any binning scheme for the ISIs. Our choice of logarithmic bins keeps the number of parameters small (proportional to $N_{\text{bins}}$), but preserves a constant temporal resolution.

The hidden-state dynamics form Poisson processes characterized by a lifetime. By esti-

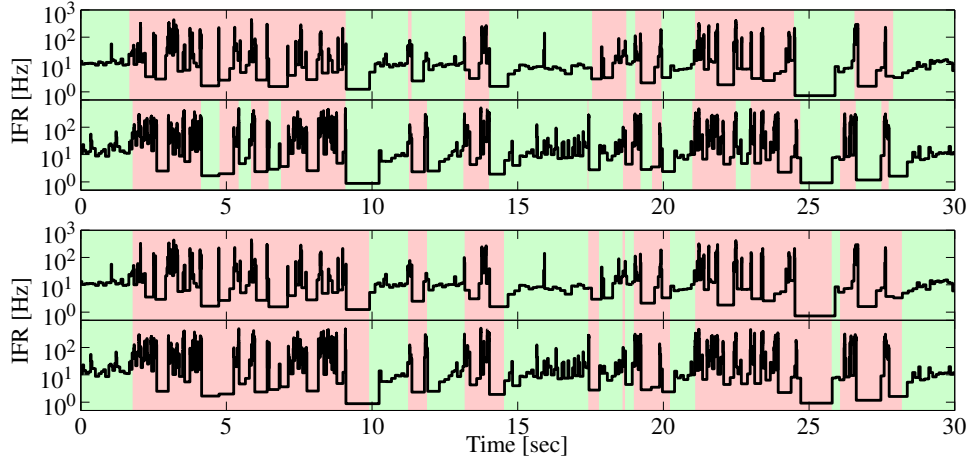

Figure 3: Shown are the instantaneous firing rate (IFR) functions of two simultaneously recorded RA neurons (at any time, the IFR corresponds to the inverse of the current ISI). The green areas show the times when in the first (awake-like) hidden state, and the red areas when in the song-like hidden state. The top two rows show the result of computing two independent models on the two neurons, whereas the bottom rows show the result of a single model.

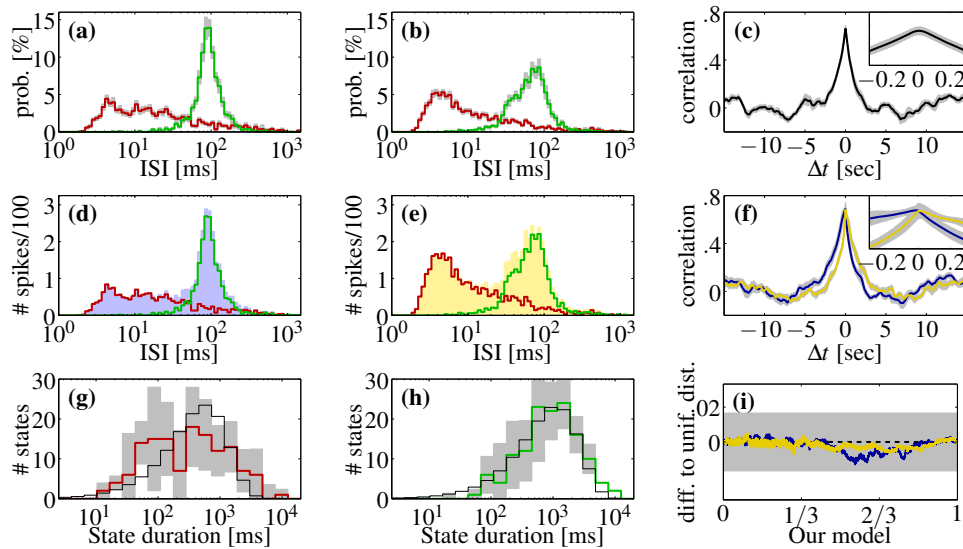

Figure 4: (a) and (b): State-conditional ISI pdfs for each of the two neurons. (d) and (e): ISI histograms (blue and yellow) for neurons 1 and 2, respectively, as well as state-conditional ISI histograms (red and green), computed as in Figure 2a. (g) and (h): State lifetime histograms for the song-like state (red) and for the awake-like state (green). Theoretical (exponential) histograms with escape rates $r_1$ and $r_2$ (fine black lines) show good agreement with the measured histograms, especially in F. (c): Correlation between state functions of the two separate models. (f): Correlation between the state functions of the combined model with separate model 1 (blue) and separate model 2 (yellow). (i): Kolmogorov-Smirnov plot after time rescaling. After transforming the ISIs, the resulting densities for both neurons remain within the 95% confidence bounds of the uniform density (gray area). In (a)–(c) and (f)–(h), Jackknife standard deviations are shown by the gray areas.

mating this lifetime, we hope it might be possible to form a link between the hidden states and the underlying physical process that governs the dynamics of switching. Despite the apparent limitation of Poisson statistics, it is a simple matter to generalize our model to hidden state distributions with long tails (e.g., power-law lifetime distributions): By cascading many hidden states into a chain (with fixed CIFs), a power-law distribution can be approximated by the combination of multiple exponentials with different lifetimes. Our code is available at `http://www.ini.unizh.ch/~rich/software/`.

**Acknowledgements**

We would like to thank Sam Roweis for advice on Hidden Markov models and Maria Minkoff for help with the manuscript. R. H. is supported by the Swiss National Science Foundation. M. D. is supported by Stiftung der Deutschen Wirtschaft.

# References

[1] Z. Nádasdy, H. Hirase, A. Czurkó, J. Csicsvári, and G. Buzsáki. Replay and time compression of recurring spike sequences in the hippocampus. *J Neurosci*, 19(21):9497–9507, Nov 1999.

[2] K. G. Thompson, D. P. Hanes, N. P. Bichot, and J. D. Schall. Perceptual and motor processing stages identified in the activity of macaque frontal eye field neurons during visual search. *J Neurophysiol*, 76(6):4040–4055, Dec 1996.

[3] R. Cossart, D. Aronov, and R. Yuste. Attractor dynamics of network UP states in the neocortex. *Nature*, 423(6937):283–288, May 2003.

[4] E. N. Brown, R. Barbieri, V. Ventura, R. E. Kass, and L. M. Frank. The time-rescaling theorem and its application to neural spike train data analysis. *Neur Comp*, 14(2):325–346, Feb 2002.

[5] J. Deppisch, K. Pawelzik, and T. Geisel. Uncovering the synchronization dynamics from correlated neuronal activity quantifies assembly formation. *Biol Cybern*, 71(5):387–399, 1994.

[6] A. M. Fraser and A. Dimitriadis. Forecasting probability densities by using hidden Markov models with mixed states. In Weigend and Gershenfeld, editors, *Time Series Prediction: Forecasting the Future and Understanding the Past*, pages 265–82. Addison-Wesley, 1994.

[7] A. Berchtold. The double chain Markov model. *Comm Stat Theor Meths*, 28:2569–2589, 1999.

[8] L. R. Rabiner. A tutorial on hidden Markov models and selected applications in speech recognition. *Proc IEEE*, 77(2):257–286, Feb 1989.

[9] R. H. R. Hahnloser, A. A. Kozhevnikov, and M. S. Fee. An ultra-sparse code underlies the generation of neural sequences in a songbird. *Nature*, 419(6902):65–70, Sep 2002.

[10] A. S. Dave and D. Margoliash. Song replay during sleep and computational rules for sensorimotor vocal learning. *Science*, 290(5492):812–816, Oct 2000.

[11] D. J. Thomson and A. D. Chave. Jackknifed error estimates for spectra, coherences, and transfer functions. In Simon Haykin, editor, *Advances in Spectrum Analysis and Array Processing*, volume 1, chapter 2, pages 58–113. Prentice Hall, 1991.

[12] A. Leonardo and M. S. Fee. Ensemble coding of vocal control in birdsong. *J Neurosci*, 25(3):652–661, Jan 2005.

[13] G. Radons, J. D. Becker, B. Dülfer, and J. Krüger. Analysis, classification, and coding of multielectrode spike trains with hidden Markov models. *Biol Cybern*, 71(4):359–373, 1994.

[14] I. Gat, N. Tishby, and M. Abeles. Hidden Markov modelling of simultaneously recorded cells in the associative cortex of behaving monkeys. *Network: Computation in Neural Systems*, 8(3):297–322, 1997.
